# Group and Topic Discovery
# from Relations and Their Attributes

**Xuerui Wang, Natasha Mohanty, Andrew McCallum**
Department of Computer Science
University of Massachusetts
Amherst, MA 01003
{xuerui,nmohanty,mccallum}@cs.umass.edu

## Abstract

We present a probabilistic generative model of entity relationships and their attributes that simultaneously discovers groups among the entities and topics among the corresponding textual attributes. Block-models of relationship data have been studied in social network analysis for some time. Here we simultaneously cluster in several modalities at once, incorporating the attributes (here, words) associated with certain relationships. Significantly, joint inference allows the discovery of topics to be guided by the emerging groups, and vice-versa. We present experimental results on two large data sets: sixteen years of bills put before the U.S. Senate, comprising their corresponding text and voting records, and thirteen years of similar data from the United Nations. We show that in comparison with traditional, separate latent-variable models for words, or Block-structures for votes, the Group-Topic model's joint inference discovers more cohesive groups and improved topics.

## 1 Introduction

The field of social network analysis (SNA) has developed mathematical models that discover patterns in interactions among entities. One of the objectives of SNA is to detect salient groups of entities. Group discovery has many applications, such as understanding the social structure of organizations or native tribes, uncovering criminal organizations, and modeling large-scale social networks in Internet services such as Friendster.com or LinkedIn.com. Social scientists have conducted extensive research on group detection, especially in fields such as anthropology and political science. Recently, statisticians and computer scientists have begun to develop models that specifically discover group memberships [5, 2, 7]. One such model is the stochastic Blockstructures model [7], which discovers the latent groups or classes based on pair-wise relation data. A particular relation holds between a pair of entities (people, countries, organizations, etc.) with some probability that depends only on the class (group) assignments of the entities. This model is extended in [4] to support an arbitrary number of groups by using a Chinese Restaurant Process prior.

The aforementioned models discover latent groups by examining only whether one or more relations exist between a pair of entities. The Group-Topic (GT) model presented in this paper, on the other hand, considers both the relations between entities and also the attributes

of the relations (e.g., the text associated with the relations) when assigning group memberships. The GT model can be viewed as an extension of the stochastic Blockstructures model [7] with the key addition that group membership is conditioned on a latent variable, which in turn is also associated with the attributes of the relation. In our experiments, the attributes of relations are words, and the latent variable represents the topic responsible for generating those words. Our model captures the *(language) attributes* associated with interactions, and uses distinctions based on these attributes to better assign group memberships.

Consider a legislative body and imagine its members forming coalitions (groups), and voting accordingly. However, different coalitions arise depending on the topic of the resolution up for a vote. In the GT model, the discovery of groups is guided by the emerging topics, and the forming of topics is shaped by emerging groups.Resolutions that would have been assigned the same topic in a model using words alone may be assigned to different topics if they exhibit distinct voting patterns. Topics may be merged if the entities vote very similarly on them. Likewise, multiple different divisions of entities into groups are made possible by conditioning them on the topics.

The importance of modeling the *language* associated with interactions between people has recently been demonstrated in the Author-Recipient-Topic (ART) model [6]. It can measure role similarity by comparing the topic distributions for two entities. However, the ART model does not explicitly discover groups formed by entities. When forming latent groups, the GT model simultaneously discovers salient topics relevant to relationships between entities—topics which the models that only examine words are unable to detect.

We demonstrate the capabilities of the GT model by applying it to two large sets of voting data: one from US Senate and the other from the General Assembly of the UN. The model clusters voting entities into coalitions and simultaneously discovers topics for word attributes describing the relations (bills or resolutions) between entities. We find that the groups obtained from the GT model are significantly more cohesive ($p$-value $< 0.01$) than those obtained from the Blockstructures model. The GT model also discovers new and more salient topics that help better predict entities' behaviors.

## 2 Group-Topic Model

The Group-Topic model is a directed graphical model that clusters entities with relations between them, as well as attributes of those relations. The relations may be either symmetric or asymmetric and have multiple attributes. In this paper, we focus on symmetric relations and have words as the attributes on relations. The graphical model representation of the model and our notation are shown in Figure 1.

Without considering the topics of events, or by treating all events in a corpus as reflecting a single topic, the simplified model becomes equivalent to the stochastic Blockstructures model [7]. Here, each event defines a relationship, *e.g.*, whether in the event two entities' group(s) behave the same way or not. On the other hand, in our model a relation may also have multiple attributes. When we consider the complete model, the dataset is dynamically divided into $T$ sub-blocks each of which corresponds to a topic. The generative process of the GT model is as right.

$$
\begin{aligned}
t_b &\sim \text{Uniform}(1/T) \\
w_{it}|\phi_t &\sim \text{Multinomial}(\phi_t) \\
\phi_t|\eta &\sim \text{Dirichlet}(\eta) \\
g_{it}|\theta_t &\sim \text{Multinomial}(\theta_t) \\
\theta_t|\alpha &\sim \text{Dirichlet}(\alpha) \\
v_{ij}^{(b)}|\gamma_{g_i g_j}^{(b)} &\sim \text{Binomial}(\gamma_{g_i g_j}^{(b)}) \\
\gamma_{gh}^{(b)}|\beta &\sim \text{Beta}(\beta).
\end{aligned}
$$

We want to perform joint inference on (text) attributes and relations to obtain topic-wise group memberships. We employ Gibbs sampling to conduct inference. Note that we adopt conjugate priors in our setting, and thus we can easily integrate out $\theta$, $\phi$ and $\gamma$ to decrease

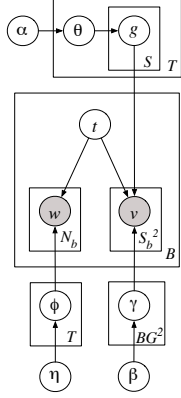

| SYMBOL | DESCRIPTION |
|---|---|
| $g_{st}$ | entity $s$'s group assignment in topic $t$ |
| $t_b$ | topic of an event $b$ |
| $w_k^{(b)}$ | the $k$th token in the event $b$ |
| $v_{ij}^{(b)}$ | entity $i$ and $j$'s group(s) behaved same (1) or differently (2) on the event $b$ |
| $S$ | # of entities |
| $T$ | # of topics |
| $G$ | # of groups |
| $B$ | # of events |
| $V$ | # of unique words |
| $N_b$ | # of word tokens in the event $b$ |
| $S_b$ | # of entities who participated in the event $b$ |

Figure 1: The Group-Topic model and notations used in this paper

the uncertainty associated with them.. In our case we need to compute the conditional distribution $P(g_{st}|\mathbf{w}, \mathbf{v}, \mathbf{g}_{-st}, \mathbf{t}, \alpha, \beta, \eta)$ and $P(t_b|\mathbf{w}, \mathbf{v}, \mathbf{g}, \mathbf{t}_{-b}, \alpha, \beta, \eta)$, where $\mathbf{g}_{-st}$ denotes the group assignments for all entities except entity $s$ in topic $t$, and $\mathbf{t}_{-b}$ represents the topic assignments for all events except event $b$. Beginning with the joint probability of a dataset, and using the chain rule, we can obtain the conditional probabilities conveniently. In our setting, the relationship we are investigating is always symmetric, so we do not distinguish $R_{ij}$ and $R_{ji}$ in our derivations (only $R_{ij}(i \le j)$ remain). Thus

$P(g_{st}|\mathbf{v}, \mathbf{g}_{-st}, \mathbf{w}, \mathbf{t}, \alpha, \beta, \eta)$

$$\propto \frac{\alpha_{g_{st}} + n_{t g_{st}} - 1}{\sum_{g=1}^{G}(\alpha_g + n_{tg}) - 1} \prod_{b=1}^{B} \left( I(t_b = t) \prod_{h=1}^{G} \frac{\prod_{k=1}^{2} \prod_{x=1}^{d_{g_{st}hk}^{(b)}} \left(\beta_k + m_{g_{st}hk}^{(b)} - x\right)}{\prod_{x=1}^{\sum_{k=1}^{2} d_{g_{st}hk}^{(b)}} \left( (\sum_{k=1}^{2}(\beta_k + m_{g_{st}hk}^{(b)})) - x\right)} \right),$$

where $n_{tg}$ represents how many entities are assigned into group $g$ in topic $t$, $c_{tv}$ represents how many tokens of word $v$ are assigned to topic $t$, $m_{ghk}^{(b)}$ represents how many times group $g$ and $h$ vote same ($k = 1$) and differently ($k = 2$) on event $b$, $I(t_b = t)$ is an indicator function, and $d_{g_{st}hk}^{(b)}$ is the increase in $m_{g_{st}hk}^{(b)}$ if entity $s$ were assigned to group $g_{st}$ than without considering $s$ at all (if $I(t_b = t) = 0$, we ignore the increase in event $b$).

$P(t_b|\mathbf{v}, \mathbf{g}, \mathbf{w}, \mathbf{t}_{-b}, \alpha, \beta, \eta)$

$$\propto \frac{\prod_{v=1}^{V} \prod_{x=1}^{e_v^{(b)}} (\eta_v + c_{t_b v} - x)}{\prod_{x=1}^{\sum_{v=1}^{V} e_v^{(b)}} \left( \sum_{v=1}^{V} (\eta_v + c_{t_b v}) - x \right)} \prod_{g=1}^{G} \prod_{h=g}^{G} \frac{\prod_{k=1}^{2} \Gamma(\beta_k + m_{ghk}^{(b)})}{\Gamma(\sum_{k=1}^{2}(\beta_k + m_{ghk}^{(b)}))},$$

where $e_v^{(b)}$ is the number of tokens of word $v$ in event $b$.

The GT model uses information from two different modalities whose likelihoods are generally not directly comparable, since the number of occurrences of each type may vary greatly. Thus we raise the first term in the above formula to a power, as is common in speech recognition when the acoustic and language models are combined.

## 3 Related Work

There has been a surge of interest in models that describe relational data, or relations between entities viewed as links in a network, including recent work in group discovery [2, 5]. The GT model is an enhancement of the stochastic Blockstructures model [7] and

| Datasets | Avg. AI for GT | Avg. AI for Baseline | $p$-value |
|:---:|:---:|:---:|:---:|
| **Senate** | 0.8294 | 0.8198 | $< .01$ |
| **UN** | 0.8664 | 0.8548 | $< .01$ |

Table 1: Average AI for GT and Baseline for both Senate and UN datasets. The group cohesion in GT is significantly better than in baseline.

the extended model of Kemp et al. [4] as it takes advantage of information from different modalities by conditioning group membership on topics. In this sense, the GT model draws inspiration from the Role-Author-Recipient-Topic (RART) model [6]. As an extension of ART model, RART clusters together entities with similar roles. In contrast, the GT model presented here clusters entities into groups based on their relations to other entities.

There has been a considerable amount of previous work in understanding voting patterns. Exploring the notion that the behavior of an entity can be explained by its (hidden) group membership, Jakulin and Buntine [3] develop a discrete PCA model for discovering groups, where each entity can belong to each of the $k$ groups with a certain probability, and each group has its own specific pattern of behaviors. They apply this model to voting data in the 108th US Senate where the behavior of an entity is its vote on a resolution. We apply our GT model also to voting data. However, unlike [3], since our goal is to cluster entities based on the similarity of their voting patterns, we are only interested in whether a pair of entities voted the same or differently, not their actual yes/no votes. This "content-ignorant" feature is similarly found in work on web log clustering [1].

## 4 Experimental Results

We present experiments applying the GT model to the voting records of members of two legislative bodies: the US Senate and the UN General Assembly. For comparison, we present the results of a baseline method that first uses a mixture of unigrams to discover topics and associate a topic with each resolution, and then runs the Blockstructures model [7] separately on the resolutions assigned to each topic. This baseline approach is similar to the GT model in that it discovers both groups and topics, and has different group assignments on different topics. However, whereas the baseline model performs inference serially, GT performs joint inference simultaneously.

We are interested in the quality of both the groups and the topics. In the political science literature, group cohesion is quantified by the *Agreement Index (AI)* [3], which, based on the number of group members that vote Yes, No or Abstain, measures the similarity of votes cast by members of a group during a particular roll call. Higher AI means better cohesion. The group cohesion using the GT model is found to be significantly greater than the baseline group cohesion under pairwise $t$-test, as shown in Table 1 for both datasets, which indicates that the GT model is better able to capture cohesive groups.

### 4.1 The US Senate Dataset

Our Senate dataset consists of the voting records of Senators in the 101st-109th US Senate (1989-2005) obtained from the Library of Congress THOMAS database. During a roll call for a particular bill, a Senator may respond *Yea* or *Nay* to the question that has been put to vote, else the vote will be recorded as *Not Voting*. We do not consider *Not Voting* as a unique vote since most of the time it is a result of a Senator being absent from the session of the US Senate. The text associated with each resolution is composed of its index terms provided in the database. There are 3423 resolutions in our experiments (we excluded roll calls that were not associated with resolutions). Since there are far fewer words than

| Economic | Education | Military Misc. | Energy |
|----------|-----------|----------------|--------|
| federal | education | government | energy |
| labor | school | military | power |
| insurance | aid | foreign | water |
| aid | children | tax | nuclear |
| tax | drug | congress | gas |
| business | students | aid | petrol |
| employee | elementary | law | research |
| care | prevention | policy | pollution |

Table 2: Top words for topics generated with the mixture of unigrams model on the Senate dataset. The headers are our own summary of the topics.

| Economic | Education + Domestic | Foreign | Social Security + Medicare |
|----------|----------------------|---------|----------------------------|
| labor | education | foreign | social |
| insurance | school | trade | security |
| tax | federal | chemicals | insurance |
| congress | aid | tariff | medical |
| income | government | congress | care |
| minimum | tax | drugs | medicare |
| wage | energy | communicable | disability |
| business | research | diseases | assistance |

Table 3: Top words for topics generated with the GT model on the Senate dataset. The topics are influenced by both the words and votes on the bills.

pairs of votes, we raise the text likelihood to the 5th power (mentioned in Section 2) in the experiments with this dataset so as to balance its influence during inference.

We cluster the data into 4 topics and 4 groups (cluster sizes are chosen somewhat arbitrarily) and compare the results of GT with the baseline. The most likely words for each topic from the traditional mixture of unigrams model is shown in Table 2, whereas the topics obtained using GT are shown in Table 3. The GT model collapses the topics **Education** and **Energy** together into **Education and Domestic**, since the voting patterns on those topics are quite similar. The new topic **Social Security + Medicare** did not have strong enough word coherence to appear in the baseline model, but it has a very distinct voting pattern, and thus is clearly found by the GT model. Thus, importantly, GT discovers topics that help predict people's behavior and relations, not simply word co-occurrences.

Examining the group distribution across topics in the GT model, we find that on the topic **Economic** the Republicans form a single group whereas the Democrats split into 3 groups indicating that Democrats have been somewhat divided on this topic. On the other hand, in **Education + Domestic** and **Social Security + Medicare**, Democrats are more unified whereas the Republicans split into 3 groups. The group membership of Senators on **Education + Domestic** issues is shown in Table 4. We see that the first group of Republicans include a Democratic Senator from Texas, a state that usually votes Republican. Group 2 (majority Democrats) includes Sen. Chafee who has been involved in initiatives to improve education, as well as Sen. Jeffords who left the Republican Party to become an Independent and has championed legislation to strengthen education and environmental protection.

Nearly all the Republican Senators in Group 4 (in Table 4) are advocates for education and many of them have been awarded for their efforts. For instance, Sen. Voinovich and Sen. Symms are strong supporters of early education and vocational education, respectively; and

| Group 1 | Group 3 | Group 4 | |
|---|---|---|---|
| 73 Republicans | Cohen (R-ME) | Armstrong (R-CO) | Brown (R-CO) |
| Krueger (D-TX) | Danforth (R-MO) | Garn (R-UT) | DeWine (R-OH) |
| **Group 2** | Durenberger (R-MN) | Humphrey (R-NH) | Thompson (R-TN) |
| 90 Democrats | Hatfield (R-OR) | McCain (R-AZ) | Fitzgerald (R-IL) |
| Chafee (R-RI) | Heinz (R-PA) | McClure (R-ID) | Voinovich (R-OH) |
| Jeffords (I-VT) | Kassebaum (R-KS) | Roth (R-DE) | Miller (D-GA) |
| | Packwood (R-OR) | Symms (R-ID) | Coleman (R-MN) |
| | Specter (R-PA) | Wallop(R-WY) | |
| | Snowe (R-ME) | | |
| | Collins (R-ME) | | |

Table 4: Senators in the four groups corresponding to **Education + Domestic** in Table 3.

| Everything Nuclear | Human Rights | Security in Middle East |
|---|---|---|
| nuclear | rights | occupied |
| weapons | human | israel |
| use | palestine | syria |
| implementation | situation | security |
| countries | israel | calls |

Table 5: Top words for topics generated from mixture of unigrams model with the UN dataset. Only text information is utilized to form the topics, as opposed to Table 6 where our GT model takes advantage of both text and voting information.

Sen. Roth has voted for tax deductions for education. It is also interesting to see that Sen. Miller (D-GA) appears in a Republican group; although he is in favor of educational reforms, he is a conservative Democrat and frequently criticizes his own party—even backing Republican George W. Bush over Democrat John Kerry in the 2004 Presidential Election.

Many of the Senators in Group 3 have also focused on education and other domestic issues such as energy, however, they often have a more liberal stance than those in Group 4, and come from states that are historically less conservative. For example, Sen. Danforth has presented bills for a more fair distribution of energy resources. Sen. Kassebaum is known to be uncomfortable with many Republican views on domestic issues such as education, and has voted against voluntary prayer in school. Thus, both Groups 3 and 4 differ from the Republican core (Group 2) on domestic issues, and also differ from each other.

We also inspect the Senators that switch groups the most across topics in the GT model. The top 5 Senators are Shelby (D-AL), Heflin (D-AL), Voinovich (R-OH), Johnston (D-LA), and Armstrong (R-CO). Sen. Shelby (D-AL) votes with the Republicans on **Economic**, with the Democrats on **Education + Domestic** and with a small group of maverick Republicans on **Foreign** and **Social Security + Medicare**. Sen. Shelby, together with Sen. Heflin, is a Democrat from a fairly conservative state (Alabama) and are found to side with the Republicans on many issues.

## 4.2 The United Nations Dataset

The second dataset involves the voting record of the UN General Assembly[1]. We focus on the resolutions discussed from 1990-2003, which contain votes of 192 countries on 931 resolutions. If a country is present during the roll call, it may choose to vote *Yes*, *No* or

| GROUP ↓ | Nuclear Nonproliferation | Nuclear Arms Race | Human Rights |
|---|---|---|---|
| | nuclear | nuclear | rights |
| | states | arms | human |
| | united | prevention | palestine |
| | weapons | race | occupied |
| | nations | space | israel |
| 1 | Brazil<br>Columbia<br>Chile<br>Peru<br>Venezuela... | UK<br>France<br>Spain<br>Monaco<br>East-Timor | Brazil<br>Mexico<br>Columbia<br>Chile<br>Peru... |
| 2 | USA<br>Japan<br>Germany<br>UK...<br>Russia... | India<br>Russia<br>Micronesia | Nicaragua<br>Papua<br>Rwanda<br>Swaziland<br>Fiji... |
| 3 | China<br>India<br>Mexico<br>Iran<br>Pakistan... | Japan<br>Germany<br>Italy...<br>Poland<br>Hungary... | USA<br>Japan<br>Germany<br>UK...<br>Russia... |
| 4 | Kazakhstan<br>Belarus<br>Yugoslavia<br>Azerbaijan<br>Cyprus... | China<br>Brazil<br>Mexico<br>Indonesia<br>Iran... | China<br>India<br>Indonesia<br>Thailand<br>Philippines... |
| 5 | Thailand<br>Philippines<br>Malaysia<br>Nigeria<br>Tunisia... | USA<br>Israel<br>Palau | Belarus<br>Turkmenistan<br>Azerbaijan<br>Uruguay<br>Kyrgyzstan... |

Table 6: Top words for topics generated from the GT model with the UN dataset as well as the corresponding groups for each topic (column). The countries listed for each group are ordered by their 2005 GDP (PPP).

*Abstain*. Unlike the Senate dataset, a country's vote can have one of three possible values instead of two. Because we parameterize *agreement* and not the votes themselves, this 3-value setting does not require any change to our model. In experiments with this dataset, we use a weighting factor 500 for text (adjusting the likelihood of text by a power of 500 so as to make it comparable with the likelihood of pairs of votes for each resolution). We cluster this dataset into 3 topics and 5 groups (chosen somewhat arbitrarily).

The most probable words in each topic from the mixture of unigrams model is shown in Table 5. For example, **Everything Nuclear** constitutes all resolutions that have anything to do with the use of nuclear technology, including nuclear weapons. Comparing these with topics generated from the GT model shown in Table 6, we see that the GT model splits the discussion about nuclear technology into two separate topics, **Nuclear Nonproliferation** (generally about countries obtaining nuclear weapons and management of nuclear waste), and **Nuclear Arms Race** (focused on the historic arms race between Russia and the US, and preventing a nuclear arms race in outer space). These two issues had drastically different voting patterns in the UN, as can be seen in the contrasting group structure for those topics in Table 6. Thus, again, the GT model is able to discover more salient topics—topics

that reflect the voting patterns and coalitions, not simply word co-occurrence alone. The countries in Table 6 are ranked by their GDP in 2005.[2]

As seen in Table 6, groups formed in **Nuclear Arms Race** are unlike the groups formed in other topics. These groups map well to the global political situation of that time when, despite the end of the Cold War, there was mutual distrust between Russia and the US with regard to the continued manufacture of nuclear weapons. For missions to outer space and nuclear arms, India was a staunch ally of Russia, while Israel was an ally of the US.

## 5    Conclusions

We introduce the Group-Topic model that jointly discovers latent groups in a network as well as clusters of attributes (or topics) of events that influence the interaction between entities. The model extends prior work on latent group discovery by capturing not only pair-wise relations between entities but also multiple attributes of the relations (in particular, words describing the relations). In this way the GT model obtains more cohesive groups as well as salient topics that influence the interaction between groups. This paper demonstrates that the Group-Topic model is able to discover topics capturing the group based interactions between members of a legislative body. The model can be applied not just to voting data, but any data having relations with attributes. We are now using the model to analyze the citations in academic papers capturing the topics of research papers and discovering research groups. The model can be altered suitably to consider other categorical, multi-dimensional, and continuous attributes characterizing relations.

### Acknowledgments

This work was supported in part by the CIIR, the Central Intelligence Agency, the National Security Agency, the National Science Foundation under NSF grant #IIS-0326249, and by the Defense Advanced Research Projects Agency, through the Department of the Interior, NBC, Acquisition Services Division, under contract #NBCHD030010. We would also like to thank Prof. Vincent Moscardelli, Chris Pal and Aron Culotta for helpful discussions.

## Footnotes

[1]http://home.gwu.edu/∼voeten/UNVoting.htm

[2] http://en.wikipedia.org/wiki/List_of_countries_by_GDP_%28PPP%29. In Table 6, we omit some countries (represented by ...) in order to show other interesting but relatively low-ranked countries (for example, Russia) in the GDP list.

## References

[1] Doug Beeferman and Adam Berger. Agglomerative clustering of a search engine query log. In *The 6th ACM SIGKDD Int. Conf. on Knowledge Discovery and Data Mining*, 2000.

[2] Indrajit Bhattacharya and Lise Getoor. Deduplication and group detection using links. In *The 10th SIGKDD Conference Workshop on Link Analysis and Group Detection (LinkKDD)*, 2004.

[3] Aleks Jakulin and Wray Buntine. Analyzing the US Senate in 2003: Similarities, networks, clusters and blocs, 2004. http://kt.ijs.si/aleks/Politics/us_senate.pdf.

[4] Charles Kemp, Thomas L. Griffiths, and Joshua Tenenbaum. Discovering latent classes in relational data. Technical report, AI Memo 2004-019, MIT CSAIL, 2004.

[5] Jeremy Kubica, Andrew Moore, Jeff Schneider, and Yiming Yang. Stochastic link and group detection. In *The 17th National Conference on Artificial Intelligence (AAAI)*, 2002.

[6] Andrew McCallum, Andres Corrada-Emanuel, and Xuerui Wang. Topic and role discovery in social networks. In *The 19th International Joint Conference on Artificial Intelligence*, 2005.

[7] Krzysztof Nowicki and Tom A.B. Snijders. Estimation and prediction for stochastic blockstructures. *Journal of the American Statistical Association*, 96(455):1077–1087, 2001.

